# Predicting response time and error rates in visual search

**Bo Chen**
Caltech
bchen3@caltech.edu

Vidhya Navalpakkam
Yahoo! Research
nvidhya@yahoo-inc.com

Pietro Perona
Caltech
perona@caltech.edu

## Abstract

A model of human visual search is proposed. It predicts both response time (RT) and error rates (RT) as a function of image parameters such as target contrast and clutter. The model is an ideal observer, in that it optimizes the Bayes ratio of target present vs target absent. The ratio is computed on the firing pattern of V1/V2 neurons, modeled by Poisson distributions. The optimal mechanism for integrating information over time is shown to be a 'soft max' of diffusions, computed over the visual field by 'hypercolumns' of neurons that share the same receptive field and have different response properties to image features. An approximation of the optimal Bayesian observer, based on integrating local *decisions*, rather than diffusions, is also derived; it is shown experimentally to produce very similar predictions to the optimal observer in common psychophysics conditions. A psychophysics experiment is proposed that may discriminate between which mechanism is used in the human brain.

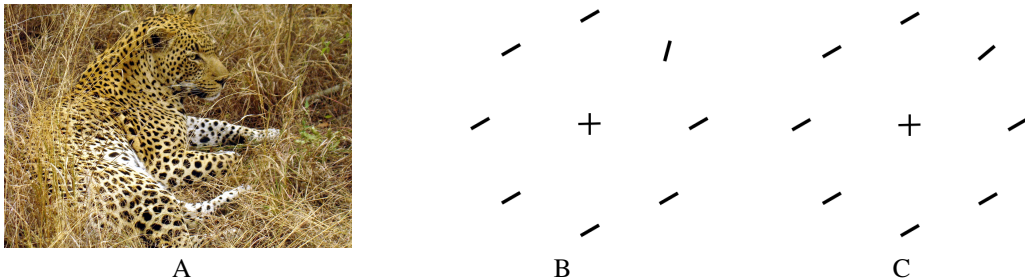

Figure 1: Visual search. (A) Clutter and camouflage make visual search difficult. (B,C) Psychologists and neuroscientists build synthetic displays to study visual search. In (B) the target 'pops out' ($\Delta\theta = 45^0$), while in (C) the target requires more time to be detected ($\Delta\theta = 10^0$) [1].

## 1 Introduction

Animals and humans often use vision to find things: mushrooms in the woods, keys on a desk, a predator hiding in tall grass. Visual search is challenging because the location of the object that one is looking for is not known in advance, and surrounding clutter may generate false alarms. The three ecologically relevant performance parameters of visual search are the two error rates (ER): false alarms (FA) and false rejects (FR), and response time (RT). The design of a visual system is crucial in obtaining low ER and RT. These parameters may be traded off by manipulating suitable thresholds [2, 3, 4].

Psychologists and physiologists have long been interested in understanding the performance and the mechanisms of visual search. In order to approach this difficult problem they present human subjects with synthetic stimuli composed of a variable number of 'items' which may include a 'target'

and multiple 'distractors' (see Fig. 1). By varying the number of items one may vary the amount of clutter; by designing different target-distractor pairs one may probe different visual cues (contrast, orientation, color, motion) and by varying the visual distinctiveness of the target vis-a-vis the distractors one may study the effect of the signal-to-noise ratio (SNR). Several studies since 1980s have investigated how RT and ER are affected by the complexity of the stimulus (number of distractors), and by target-distractor discriminability with different visual cues. One early observation is that when the target and distractor features are widely separated in feature space (e.g., red target among green distractors), the target 'pops out'. In these situations the ER is nearly zero, and the slope of RT vs. setsize is flat, i.e., RT to find the target is independent of number of items in the display [1]. Decreasing the discriminability between the target and distractor increases error rates, and increases the slope of RT vs. setsize [5]. Moreover, it was found that the RT for displays with no target is longer than where the target is present (see review in [6]). Recent studies investigated the shape of RT distributions in visual search [7, 8].

Neurophysiologically plausible models have been recently proposed to predict RTs in visual discrimination tasks [9] and various other 2AFC tasks [10] at a single spatial location in the visual field. They are based on sequential tests of statistical hypotheses (target present vs target absent) [11] computed on the response of stimulus-tuned neurons [2, 3]. We do not yet have satisfactory models for explaining RTs in visual search, which is harder as it involves integrating information across several locations across the visual field, as well as time. Existing models predicting RT in visual search are either qualitative (e.g. [12]) or descriptive (e.g., the drift-diffusion model [13, 14, 15]), and do not attempt to predict experimental results with new set sizes, target and distractor settings.

We propose a Bayesian model of visual search that predicts both ER and RT. Our study makes a number of contributions. First, while visual search has been modeled using signal-detection theory to predict ER [16], our model is based on neuron-like mechanisms and predicts both ER and RT. Second, our model is an optimal observer, given a physiologically plausible front-end of the visual system. Third, our model shows that in visual search the optimal computation is *not* a diffusion, as one might believe by analogy with single-location discrimination models [17, 18], rather, it is a 'softmax' nonlinear combination of locally-computed diffusions. Fourth, we study a physiologically parsimonious approximation to the optimal observer, we show that it is almost optimal when the characteristics of the task are known in advance and held constant, and we explore whether there are psychophysical experiments that could discriminate between the two models.

Our model is based on a number of simplifying assumptions. First, we assume that stimulus items are centered on cortical hypercolumns [19] and at locations where there is no item neuronal firing is negligible. Second, retinal and cortical magnification [19] are ignored, since psychophysicists have developed displays that sidestep this issue (by placing items on a constant-eccentricity ring as shown in Fig 1). Third, we do not account for overt and covert attentional shifts. Overt attentional shifts are manifested by saccades (eye motions), which happen every 200ms or so. Since the post-decision motor response to a stimulus by pressing a button takes about 250-300ms, one does not need to worry about eye motions when response times are shorter than 500ms. For longer RTs, one may enforce eye fixation at the center of the display so as to prevent overt attentional shifts. Furthermore, our model explains serial search without the need to invoke covert attentional shifts [20] which are difficult to prove neurophysiologically.

## 2  Target discrimination at a single location with Poisson neurons

We first consider probabilistic reasoning at one location, where two possible stimuli may appear. The stimuli differ in one respect, e.g. they have different orientations $\theta^{(1)}$ and $\theta^{(2)}$. We will call them distractor (D) and target (T), also labeled $C = 1$ and $C = 2$ (call $c \in \{1, 2\}$ the generic value of $C$). Based on the response of $N$ neurons (a hypercolumn) we will decide whether the stimulus was a target or a distractor. Crucially, a decision should be reached as soon as possible, i.e. as soon as there is sufficient evidence for T or D [11].

Given the evidence $\mathcal{T}$ (defined further below in terms of the neurons' activity) we wish to decide whether the stimulus was of type 1 or 2. We may do so when the probability $P(C = 1|\mathcal{T})$ of the stimulus being of type 1 given the observations in $\mathcal{T}$ exceeds a given threshold $T_1$ ($T_1 = 0.99$). We may instead decide in favor of $C = 2$ e.g. when $P(C = 1|\mathcal{T}) < T_2$ (e.g. $T_2 = 0.01$). If

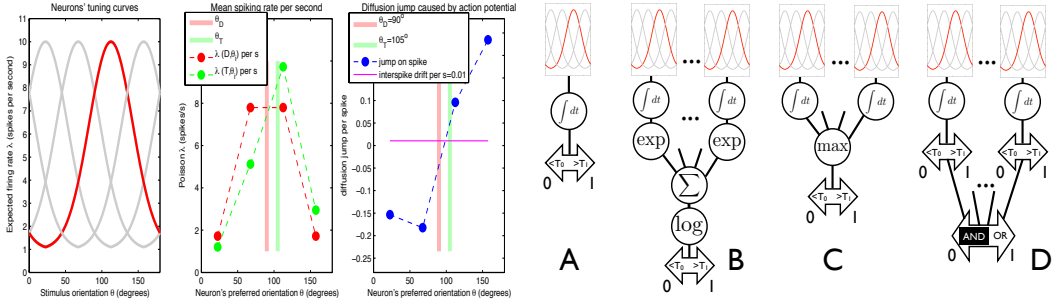

Figure 2: (Left three panels) Model of a hypercolumn in V1/V2 cortex composed of four orientation-tuned neurons (our simulations use 32). The left panel shows the neurons' tuning curve $\lambda(\theta)$ representing the expected Poisson firing rate when the stimulus has orientation $\theta$. The middle plot shows the expected firing rate of the population of neurons for two stimuli whose orientation is indicated with a red (distractor) and green (target) vertical line. The third plot shows the step-change in the value of the diffusion when an action potential is registered from a given neuron. (Right panel) Diagram of the decision models. (A) One-location Bayesian observer. The action potentials of a hypercolumn of neurons (top) are integrated in time to produce a diffusion. When the diffusion reaches either an upper bound $T_1$ or a lower bound $T_0$ the decision is taken that either the target is present (1) or the target is absent (0). (B–D) Multi-location ideal Bayesian observer. (B) While not a diffusion, it may be seen as a 'soft maximum' combination of local diffusions: the local diffusions are first exponentiated, then averaged; the log of the result is compared to two thresholds to reach a decision. (C) The 'Max approximation' is a simplified approximation of the ideal observer, where the maximum of local diffusions replaces a soft-maximum. (D) Equivalently, in the Max approximation decisions are reached locally and combined by logical operators. The white AND in a dark field indicates inverted AND of multiple inverted inputs.

$P(C = 1|\mathcal{T}) \in (T_2, T_1)$ we will wait for more evidence. Thus, we need to compute $P(C = 1|\mathcal{T})$:

$$\Pr(C = 1|\mathcal{T}) = \frac{1}{1 + \frac{P(C=2|\mathcal{T})}{P(C=1|\mathcal{T})}} = \frac{1}{1 + R(\mathcal{T})\frac{P(C=2)}{P(C=1)}}$$

$$\text{where} \quad R(\mathcal{T}) = \frac{P(\mathcal{T}|C=2)}{P(\mathcal{T}|C=1)} = \frac{P(C=2|\mathcal{T})}{P(C=1|\mathcal{T})}\frac{P(C=1)}{P(C=2)} \tag{1}$$

where $P(C = 1) = 1 - P(C = 2)$ is the prior probability of $C = 1$. Thus, it is equivalent to take decisions by thresholding $\log R(\mathcal{T})$[1]; we will elaborate on this in Sec. 3.

We will model the firing rate of the neurons with a Poisson pdf: the number $n$ of action potentials that will be observed during one second is distributed as $P(n|\lambda) = \lambda^n e^{-\lambda}/n!$. The constant $\lambda$ is the expectation of the number of action potentials per second. Each neuron $i \in \{1, \ldots, N\}$ is tuned to a different orientation $\theta_i$; for the sake of simplicity we will assume that the width of the tuning curve is the same for all neurons; i.e. each neuron $i$ will respond to stimulus $c$ with expectation $\lambda_c^i = f(|\theta^{(c)} - \theta_i|)$ (in spikes per second) which are determined by the distance between the neuron's preferred orientation $\theta_i$ and by the stimulus orientation $\theta^{(c)}$.

Let $\mathcal{T}_i = \{t_k^i\}$ be the set of action potentials from neuron $i$ produced starting at $t = 0$ and until the end of the observation period $t = T$. Indicate with $\mathcal{T} = \{t_k\} = \bigcup_i \mathcal{T}_i$ the complete set of action potentials from all neurons (where the $t_k$ are sorted). We will indicate with $i(k)$ the index of the neuron who fired the action potential at time $t_k$. Call $I_k = (t_k \ t_{k+1})$ the intervals of time in between action potentials, where $I_0 = (0 \ t_1)$. These intervals are *open* i.e. they do not contain the boundaries, hence they do not contain the action potentials.

The signal coming from the neurons is thus a concatenation of 'spikes' and 'intervals', and the interval $(0, T)$ may be viewed as the union of instants $t_k$ and open intervals $(t_k, t_{k+1})$. i.e. $(0, T) = I_0 \bigcup t_1 \bigcup I_1 \bigcup t_2 \bigcup \cdots$

Since the spike trains $\mathcal{T}_i$ and $\mathcal{T}$ are Poisson processes, once we condition on the class of the stimulus the spike times are independent. This implies that: $P(\mathcal{T}|C = c) = \Pi_k P(I_k|C = c)P(t_k|C = c)$. This may be proven by dividing up $(0, T)$ into smaller and smaller intervals and taking the limit for

[1]We use base 10 for all our logarithms and exponentials, i.e. $\log(x) \equiv \log_{10}(x)$ and $\exp(x) \equiv 10^x$.

the size of the intervals going to zero. The intervals containing action potentials converge to the $t_i$ and the intervals not containing action potentials may be merged into the intervals $I_i$.

Let's analyze separately the log likelihood ratio for the intervals and for the spikes.

**Diffusion drift during the intervals.** During the intervals no neuron spiked. The ratio therefore is computed as a function of the Poissons $P(n = 0|\lambda)$ when the spike count $n$ is zero. The Poisson expectation has to be multiplied by the time-length of the interval; call $\Delta t_k = t_{k+1} - t_k$ the length of the interval $I_k$. Assuming that the neurons $i = 1, \ldots, N$ are independent we obtain:

$$\log R(I_k) = \log \frac{P(n = 0|C = 2, t \in I_k)}{P(n = 0|C = 1, t \in I_k)} = \log \frac{\Pi_{i=1}^N P(n = 0|\lambda_2^i \Delta t_k)}{\Pi_{i=1}^N P(n = 0|\lambda_1^i \Delta t_k)} = \Delta t_k \sum_{i=1}^N (\lambda_1^i - \lambda_2^i)$$

(2)

Thus, during the time-intervals where no action potential is observed, the diffusion drifts linearly with a slope equal to the sum over all neurons of the difference between the expected firing rate with stimulus 1 and the expected firing rate with stimulus 2.

Notice that if there are neurons that fire equally well to targets and distractors, and if the population of neurons is large and made of neurons whose tuning curve's shape is identical and whose preferred orientation $\theta_i$ is regularly spaced, then $\sum_i \lambda_1^i \approx \sum_i \lambda_2^i$, thus the diffusion has drift with slope close to zero and the drift term may be ignored. In this case intervals carry no information.

**Diffusion jump at the action potentials.** If the neurons are uncorrelated, then the probability of two or more action potentials happening at the same time is zero. Thus, at any time $t_k$ there is only one action potential from one neuron. We can compute the likelihood ratio by taking a limit for the length $\delta t$ of the interval $t \in (t_k - \delta t/2, t_k + \delta t/2)$ going to zero. As seen in the previous section, the contribution from the neurons who did not register a spike is $\delta t(\lambda_1^i - \lambda_2^i)$ and goes to zero as $\delta t \to 0$. Thus we are only left with the contribution of the neuron $i(k)$ whose spike happened at time $t_k$.

$$\log R(t_k) = \lim_{\delta t \to 0} \log \frac{P(n = 1|\lambda_2^{i(k)} \delta t)}{P(n = 1|\lambda_1^{i(k)} \delta t_k)} = \lim_{\delta t \to 0} \log \frac{(\lambda_2^{i(k)} \delta t)^1 e^{-\lambda_2^{i(k)} \delta t}}{(\lambda_1^{i(k)} \delta t)^1 e^{-\lambda_1^{i(k)} \delta t}} = \log \frac{\lambda_2^{i(k)}}{\lambda_1^{i(k)}}$$

(3)

As a result, at each action potential $t_k$ the diffusion jumps by an amount that is the log of the ratio of the expected firing rate of the neuron $i(k)$'s response to target vs distractor. Thus:

1. Neurons that are equally tuned to target and distractor, whether they respond much or not, will not contribute to the diffusion, while neurons whose response is very different for target and distractor will contribute substantially to the diffusion.

2. A larger number of neurons will produce more action potentials and thus a faster action-potential-driven drift in the diffusion.

**Diffusion overall.** Given the analysis presented above:

$$\log R(\mathcal{T}) = \sum_k \Delta t_k \sum_i (\lambda_1^i - \lambda_2^i) + \sum_k \log \frac{\lambda_2^{i(k)}}{\lambda_1^{i(k)}} = |\mathcal{T}| \sum_i (\lambda_1^i - \lambda_2^i) + \sum_k \log \frac{\lambda_2^{i(k)}}{\lambda_1^{i(k)}}$$

(4)

Ignoring diffusion during the intervals, the diffusion at a single location where the stimulus is of type $c$ can be described as:

$$\log R(\mathcal{T}) \sim \sum_{i=1}^N (\log \frac{\lambda_2^i}{\lambda_1^i}) Poiss(\lambda_c^i | \mathcal{T}|)$$

(5)

$$\mathbb{E}[\log R(\mathcal{T})] = a_c |\mathcal{T}|, \mathbb{V}[\log R(\mathcal{T})] = b_c^2 |\mathcal{T}|$$

(6)

where $Poiss(\lambda)$ denotes a Poisson distributed variable with mean $\lambda$, $a_c \equiv \sum_{i=1}^N (\log \frac{\lambda_2^i}{\lambda_1^i}) \lambda_c^i$ and $b_c^2 \equiv \sum_{i=1}^N (\log \frac{\lambda_2^i}{\lambda_1^i})^2 \lambda_c^i$. The mean and variance of the diffusion grows linearly with time.

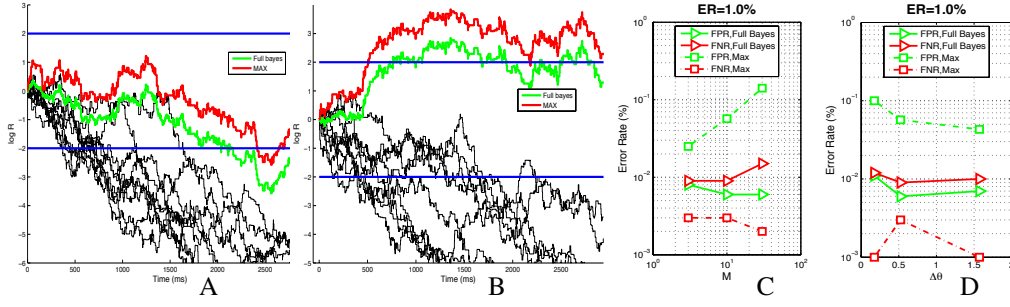

Figure 3: (A) Diffusions realized at 10 spatial locations when the target is absent (black). The ideal observer Bayes ratio is shown in green, the max-model approximation is shown in red. Thresholds $\Theta_1 = -2, \Theta_2 = 2$ are shown, which produce 1% error rates in the ideal observer. (B) Target present case. Notice that the decision takes longer when the target is absent (see also Fig. 4). (C) Error rates vs. number of items and (D) vs target contrast when decision thresholds are held constant. Decision thresholds were chosen to obtain 5% error rates in the condition $M = 10, \Delta\theta = \pi/6$. As we change target contrast and the number of targets the optimal observer has constant error rates, while the Max approximation produces variable error rates. Testing human subjects with a mix of stimuli with different values of $M$ and $\Delta\theta$ may prevent them from adjusting decision thresholds between stimuli; thus, one would expect constant error rates if the visual system uses the ideal observer and variable error rates if it uses the Max approximation.

## 3  Visual search: detection across $M$ locations with Poisson neurons

We now consider the case with $M$ locations with $N$ Poisson neurons each. At each location we may either have a target T or a distractor D. In the whole display we have two hypotheses: no target ($C = 1$) or one target at a location $l$ ($C = 2$). The second hypothesis may be broken up into the target being at any of $M$ locations $l$. Because of this, the numerator of the likelihood ratio is the sum of $M$ terms. The Bayesian observer must integrate the action potentials from each unit to a central unit that computes the posterior beliefs. The multi-location Bayesian observer may be computed by observing that the target-present event is the union of the target-present events in each one of the locations, while the target absent event implies that each location has no target. Thus, the likelihood can be computed as the weighted sum of local likelihoods at each location in the display.

We assume that

1. The likelihood at each location is independent from the rest when the stimulus type at that location is known; i.e. $P(\mathcal{T}|C^l, \forall l) = \prod_l P(\mathcal{T}^l|C^l)$ .
2. The target, if present, is equally likely to occur at any location in the display; i.e. $\forall l, P(C^l = 2, C^{\bar{l}} = 1|C = 2) = 1/M$.

Calling $l$ a location and $\bar{l}$ the complement of that location (i.e. all locations but $l$) we have:

$$P(\mathcal{T}|C = 1) = \prod_{l=1}^{M} P(\mathcal{T}^l|C^l = 1)$$

$$P(\mathcal{T}|C = 2) = \sum_{l=1}^{M} P(\mathcal{T}|C^l = 2, C^{\bar{l}} = 1)P(C^l = 2, C^{\bar{l}} = 1|C = 2)$$

$$= \frac{1}{M}(\prod_{l=1}^{M} P(\mathcal{T}^l|C^l = 1))\sum_{l=1}^{M} R^l(\mathcal{T}^l)$$

$$\log R_{tot}(\mathcal{T}) = \log \frac{P(\mathcal{T}|C = 2)}{P(\mathcal{T}|C = 1)} = \log \frac{1}{M} \sum_{l=1}^{M} R^l(\mathcal{T}^l) = \log \frac{1}{M} \sum_{l=1}^{M} \exp(\log R^l(\mathcal{T}^l)) \quad (7)$$

Eqn. 7 tells us two things:

1. The process $\log R_{tot}$ *is not* a diffusion, in that $\log R_{tot}$ at time $t+1$ can not be computed by incrementing its value at time $t$ by a term that depends only on the interval $(t, t+1)$.

2. The process $\log R_{tot}$ may be computed easily from the local diffusions $\log R^l(\mathcal{T}^l)$ (in Sec. 4 we find an approximation that has a natural neural implementation).

Now that we know how to compute $\log R(\mathcal{T})$ for single and multi-location Bayesian observer, we may take our decision by thresholding $\log R(\mathcal{T})$ (Eqn. 1). Specifically, we choose separate thresholds for making the target absent and the target present decision, and adjusted the thresholds based on tolerance levels for the false positive rate (FPR) and the false negative rate (FNR). We keep accumulating evidence until either decision can be made.

The relationship between FPR, FNR and the two thresholds can be derived using analysis similar to [11]. When $\log R_{tot}(\mathcal{T})$ reaches the target present threshold ($\Theta_2$), with probability $P(C = 2|\mathcal{T})$, the target is present and with probability $P(C = 1|\mathcal{T})$ the target is absent, i.e. $FPR = P(C = 1|\mathcal{T})$ and $1 - FNR = P(C = 2|\mathcal{T})$. We have:

$$\Theta_2 = \log R_{tot}(\mathcal{T}) = \log \frac{P(C = 2|\mathcal{T})}{P(C = 1|\mathcal{T})} = \log \frac{1 - FNR}{FPR} \qquad (8)$$

Similarly, when $\log R(\mathcal{T})$ reaches the target absent threshold ($\Theta_1$), we have:

$$\Theta_1 = \log R_{tot}(\mathcal{T}) = \log \frac{P(C = 2|\mathcal{T})}{P(C = 1|\mathcal{T})} = \log \frac{FPR}{1 - FNR} \qquad (9)$$

Therefore, given desired FPR and FNR, we can analytically compute the upper and lower thresholds for the Full Bayesian model using Eqn. 8 and 9.

## 4 Max approximation

An alternative, more economic, approach to full Bayesian decision is to approximate the global belief using the largest local diffusion and suppress the rest. This is because, in the limit where $|\mathcal{T}|$ is large, the diffusion at the location where the target is present will dominate over the other diffusions and thus it is a good approximation of the sum in Eq. 7. We will call this approach "max approximation" and also "Max model". In this scheme, at each location a diffusion based on the local Bayesian observer is computed. If any location 'detects' a target, then a target is declared. If all locations detect a distractor, then the 'no target' condition is declared. This may not be the optimal method, but it has the advantage of requiring only two low-frequency communication lines between each location and the central decision unit. Equivalently, the max approximation can be implemented by computing the maximum of the local diffusions and comparing it to an adjusted high and a low threshold for target present/absent decision (see Fig. 2).

More specifically, let $l^*$ denote the location of maximum diffusion in the display, and $\log R^{l^*}$ denote the maximum diffusion (i.e., $\log R^{l^*} = \max_{l=1}^{M} \log R^l(\mathcal{T}^l)$). From eqn 7 we know that the global likelihood ratio is the average of the local likelihood ratios, and equivalently, the log likelihood ratio is the *soft-max* of the local diffusions:

$$\log R_{tot}(\mathcal{T}) = \log \left( \frac{1}{M} \sum_{l=1}^{M} \exp \left( \log R^l(\mathcal{T}^l) \right) \right)$$

$$= \log R^{l*} + \log \left( \frac{1}{M} (1 + \sum_{l \neq l^*} \exp(\log R^l - \log R^{l^*})) \right) \qquad (10)$$

**Target present** – When the target is present in the display, if the target is different from the distractor, the diffusion at the target's location will frequently become much higher than at other locations, and the terms corresponding to $\frac{R^l}{R^{l^*}}$ may be safely ignored. Thus, the total log likelihood ratio may be approximated by the maximum local diffusions minus a constant:

$$\log R_{tot} \approx \log R^{l^*} - \log M \qquad \text{if } R^l << R^{l^*} \qquad (11)$$

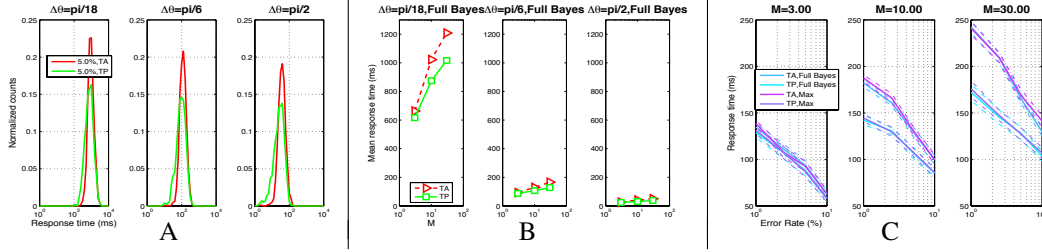

Figure 4: (A) Histogram of response-times (RT) when the target is present (green) and when the target is absent (red) for $M = 10$ for different values of target contrast ($\Delta\theta$). Response times are longer when the contrast is smaller (see Fig. 1). Also, they are longer when the target is absent (see Fig. 3). Notice that the response times have a Gaussian-like distribution when time is plotted on a log scale, and the width of the distribution does not change significantly as the difficulty of the task changes; thus, the mean and median response time are equivalently informative statistics of RT. (B) Mean RT as a function of the number $M$ of items for different values of target contrast; the curves appear linear as a function of $\log M$ [21]. Notice that RT slope is almost zero ('parallel search') when the target has high contrast, while when target contrast is low RT increases significantly with $M$ ('serial search') [1]. The response times observed using the Max approximation are almost identical to those obtained with the ideal observer. (C) Error vs. RT tradeoff curves obtained by changing systematically the value of the decision threshold. The mean RT $\pm\sigma$ is shown. Ideal bayesian observer (blue) and Max approximation (cyan) are almost identical indicating that the Max approximation's performance is almost as good as that of the optimal observer.

From Eqn. 5 and 6 we know that the difference in diffusion value between the target location and the distractor location grows linearly in time. Thus, the longer the process lasts, the better the approximation. Conversely, when $t = |\mathcal{T}|$ is small, the approximation is unreliable, and a different approximation term must be introduced (see supplementary material[2] for derivation):

$$\log R_{tot} \approx \log R^{l^*} - \left( a_2 t + \log(\frac{1}{M} + \frac{(M-1)}{M} \exp((a_1 - a_2 + \frac{b_1^2 + b_2^2}{2})t)) \right) \quad \text{if } R^l \approx R^{l^*}$$
(12)

**Target absent** – When the target is absent in the display, the value of all the local diffusions at time $t$ will be distributed according to the same density. According to Eqn. 6, the standard deviation grows as $\sqrt{t}$, hence the expected value of $\log R^{l^*} - \log R^l$ is monotonically increasing. When this expected difference is large enough, we can make the same approximation as Eqn. 11:

$$\log R_{tot} \quad \approx \quad \log R^{l^*} - \log M \qquad \text{if } R^l << R^{l^*}$$
(13)

On the other hand, when $|\mathcal{T}|$ is small, we resort to another approximation (see supplementary material for derivation):

$$\log R_{tot} \approx \log R^{l^*} - \mu_M b_1 \sqrt{t} + \frac{b_1^2 t}{2} - \frac{1}{2}\log(\frac{\exp(b^2 t) + M - 1}{M}) \quad \text{if } R^l \approx R^{l^*}$$
(14)

where $\mu_M \equiv M \int_{-\infty}^{\infty} z\Phi^{M-1}(z)\mathcal{N}(z)dz$, and $\mathcal{N}(z)$ and $\Phi(z)$ denote the pdf and cdf of normal distribution.

Since the max diffusion does not represent the global log likelihood ratio, its thresholds can not be computed directly from the error rates. Nonetheless we can first compute analytically the thresholds for the Bayesian observer (Eqn. 8 and 9), and adjust them based on the approximations stated above (Eqn. 11, 12, 13 and 14). Finally, we threshold the maximum local diffusion $\log R^{l^*}$ with respect to the adjusted upper and lower threshold to make our decision.

## 5 Experiments

**Experiment 1. - Overall model predictions.** In this experiment we explore the model's prediction of response time over a series of interesting conditions. The default parameters are the number of

neurons per location $N = 32$, the tuning width of each neuron $= \pi/8$, the maximum expected firing rate ($\lambda = 10$ action potentials per second) and minimum expected firing rate ($\lambda = 1$ a.p./s) of a neuron, which reflects the signal-to-noise ratio of the neuron's tuning curves, the number of items (locations) in the display $M = 10$ and the stimulus contrast $\Delta\theta = \pi/6$. Both $M$ and $\Delta\theta$ refers to the display, while the other parameters refer to the brain. We will focus on how predictions change when the display parameters are changed over a set of discrete settings: $M \in \{3, 10, 30\}$ and $\Delta\theta \in \{\pi/18, \pi/6, \pi/2\}$. For each setting of the parameters, we simulate the bayesian and the max model for 1000 runs. The length of simulation is set to a large value (4 seconds) to make sure that all decisions are made before the simulation terminates.

We are also interested in the trade-off between RT and ER $\eta$ for $\eta = \{1\%, 5\%, 10\%\}$. For each $\eta$ we search for the best pair of upper and lower thresholds that achieve $FNR \approx \mathrm{FPR} \approx \eta$. We search over the interval $[0\ 3.5]$ for the optimal upper threshold and over $[-3.5\ 0]$ for the optimal lower threshold (an upper threshold of 3.5 corresponds to a FPR of $0.03\%$). The search is conducted exhaustively over an $[80 \times 80]$ discretization of the joint space of the thresholds. We record the response time distributions for all parameter settings and for all values of $\eta$ (Fig. 4).

**Experiment 2. - Conditions where Bayesian and Max models differ maximally**   In this experiment we test the robustness of Bayesian and Max models with respect to a fixed threshold. For a Bayesian observer, the thresholds yielding a given error rate can be computed exactly independent of the display (Eqn. 9 and 8). On the contrary, in order for the max model to achieve the equivalent performance, its threshold must be adjusted differently depending on the number of items $M$ and the target contrasts $\Delta\theta$ (Eqn. 11-14). As a result, if a constant threshold is used for all conditions, we would expect the Bayesian observer ER to be roughly constant, whereas the Max model would have considerable ER variability. The error rates are shown in Fig. 3 as we vary $M$ and $\Delta\theta$. The threshold is set as the optimal threshold that produces $5\%$ error for the Bayesian observer at a single location $M = 1$ and with $\Delta\theta = \pi/18$.

# 6   Discussion and conclusions

We presented a Bayesian ideal observer model of visual search. To the best of our knowledge, this is the first model that can predict the statistics of both response times (RT) and error rates (ER) purely from physiologically relevant constants (number, tuning width, signal-to-noise ratio of cortical mechanisms) and from image parameters (target contrast and number of distractors). Neurons are modeled as Poisson units and the model has only four free parameters: the number of neurons per hypercolumn, the tuning width of their response curve, the maximum and the minimum firing rate of each neuron. The model predicts qualitatively the main phenomena that are observed in visual search: serial vs. parallel search [1], the Gaussian-like shape of the response time histograms in log time [7] and the faster response times when the target is present [3]. The model is easily adaptable to predictions involving multiple targets, different image features and conjunction of features.

Unlike the case of binary detection/decision, the ideal observer may not be implemented by a diffusion. However, it may be implemented using a precisely defined 'soft-max' combination of diffusions, each one of which is computed at a different location across the visual field. We discuss an approximation of the ideal observer, the Max model, which has two natural and simple implementations in neural hardware. The Max model is found experimentally to have a performance that is very close to that of the ideal observer when the task parameters do not change.

We explored whether any combinations of target contrast and number of distractors would produce significantly different predictions of the ideal observer vs the Max model approximation and found none in the case where the visual system can estimate decision thresholds in advance. However, our simulations predict different error rates when interleaving images containing diverse contrast levels and distractor numbers.

**Acknowledgements:**   We thank the three anonymous referees for many insightful comments and suggestions; thanks to M. Shadlen for a tutorial discussion on the history of discrimination models. This research was supported by the California Institute of Technology.

## Footnotes

[2]http://vision.caltech.edu/~bchen3/nips2011/supplementary.pdf

# References

[1] A.M. Treisman and G. Gelade. A feature-integration theory of attention. *Cognitive psychology*, 12(1):97–136, 1980.

[2] W.T. Newsome, K.H. Britten, and J.A. Movshon. Neuronal correlates of a perceptual decision. *Nature*, 341(6237):52–54, 1989.

[3] P. Verghese. Visual search and attention:: A signal detection theory approach. *Neuron*, 31(4):523–535, 2001.

[4] Vidhya Navalpakkam and Laurent Itti. Search goal tunes visual features optimally. *Neuron*, 53(4):605–17, Feb 2007.

[5] J. Duncan and G.W. Humphreys. Visual search and stimulus similarity. *Psychological review*, 96(3):433, 1989.

[6] J.M. Wolfe. *Attention (Ed. H. Pashler)*, chapter Visual Search, pages 13–73. University College London Press, London, U.K., 1998.

[7] J.M. Wolfe, E.M. Palmer, and T.S. Horowitz. Reaction time distributions constrain models of visual search. *Vision research*, 50(14):1304–1311, 2010.

[8] E.M. Palmer, T.S. Horowitz, A. Torralba, and J.M. Wolfe. What are the shapes of response time distributions in visual search? *Journal of Experimental Psychology: Human Perception and Performance*, 37(1):58, 2011.

[9] Jeffrey M Beck, Wei Ji Ma, Roozbeh Kiani, Tim Hanks, Anne K Churchland, Jamie Roitman, Michael N Shadlen, Peter E Latham, and Alexandre Pouget. Probabilistic population codes for bayesian decision making. *Neuron*, 60(6):1142–52, Dec 2008.

[10] R. Bogacz, E. Brown, J. Moehlis, P. Holmes, and J.D. Cohen. The physics of optimal decision making: A formal analysis of models of performance in two-alternative forced-choice tasks. *Psychological Review*, 113(4):700, 2006.

[11] A. Wald. Sequential tests of statistical hypotheses. *The Annals of Mathematical Statistics*, 16(2):117–186, 1945.

[12] M.M. Chun and J.M. Wolfe. Just say no: How are visual searches terminated when there is no target present? *Cognitive Psychology*, 30(1):39–78, 1996.

[13] R. Ratcliff. A theory of memory retrieval. *Psychological Review*, 85(2):59–108, 1978.

[14] Philip L Smith and Roger Ratcliff. Psychology and neurobiology of simple decisions. *Trends Neurosci*, 27(3):161–8, Mar 2004.

[15] Roger Ratcliff and Gail McKoon. The diffusion decision model: theory and data for two-choice decision tasks. *Neural Comput*, 20(4):873–922, Apr 2008.

[16] D.G. Pelli. Uncertainty explains many aspects of visual contrast detection and discrimination. *JOSA A*, 2(9):1508–1531, 1985.

[17] R. Ratcliff. A theory of order relations in perceptual matching. *Psychological Review*, 88(6):552, 1981.

[18] Joshua I Gold and Michael N Shadlen. The neural basis of decision making. *Annu Rev Neurosci*, 30:535–74, 2007.

[19] R.L. De Valois and K.K. De Valois. *Spatial vision*. Oxford University Press, USA, 1990.

[20] MI Posner, Y. Cohen, and RD Rafal. Neural systems control of spatial orienting. *Philosophical Transactions of the Royal Society of London. B, Biological Sciences*, 298(1089):187, 1982.

[21] W.E. Hick. On the rate of gain of information. *Quarterly Journal of Experimental Psychology*, 4(1):11–46, 1952.

